# Discovering Viewpoint-Invariant Relationships That Characterize Objects

**Richard S. Zemel** and **Geoffrey E. Hinton**
Department of Computer Science
University of Toronto
Toronto, ONT M5S 1A4

## Abstract

Using an unsupervised learning procedure, a network is trained on an ensemble of images of the same two-dimensional object at different positions, orientations and sizes. Each half of the network "sees" one fragment of the object, and tries to produce as output a set of 4 parameters that have high mutual information with the 4 parameters output by the other half of the network. Given the ensemble of training patterns, the 4 parameters on which the two halves of the network can agree are the position, orientation, and size of the whole object, or some recoding of them. After training, the network can reject instances of other shapes by using the fact that the predictions made by its two halves disagree. If two competing networks are trained on an unlabelled mixture of images of two objects, they cluster the training cases on the basis of the objects' shapes, independently of the position, orientation, and size.

## 1  INTRODUCTION

A difficult problem for neural networks is to recognize objects independently of their position, orientation, or size. Models addressing this problem have generally achieved viewpoint-invariance either through a separate normalization procedure or by building translation- or rotation-invariance into the structure of the network. This problem becomes even more difficult if the network must learn to perform viewpoint-invariant recognition without any supervision signal that indicates the correct viewpoint, or which object is which during training.

In this paper, we describe a model that is trained on an ensemble of instances of the same object, in a variety of positions, orientations and sizes, and can then recognize

new instances of that object. We also describe an extension to the model that allows it to learn to recognize two different objects through unsupervised training on an unlabelled mixture of images of the objects.

## 2    THE VIEWPOINT CONSISTENCY CONSTRAINT

An important invariant in object recognition is the fixed spatial relationship between a rigid object and each of its component features. We assume that each feature has an intrinsic reference frame, which can be specified by its *instantiation parameters*, i.e., its position, orientation and size with respect to the image. For a rigid object and a particular feature of that object, there is a *fixed* viewpoint-independent transformation from the feature's reference frame to the object's. Given the instantiation parameters of the feature in an image, we can use the transformation to predict the object's instantiation parameters. The *viewpoint consistency constraint* (Lowe, 1987) states that all of the features belonging to the same rigid object should make consistent predictions of the object's instantiation parameters. This constraint has been played an important role in many shape recognition systems (Roberts, 1965; Ballard, 1981; Hinton, 1981; Lowe, 1985).

### 2.1    LEARNING THE CONSTRAINT: SUPERVISED

A recognition system that *learns* this constraint is TRAFFIC (Zemel, Mozer and Hinton, 1989). In TRAFFIC, the constraints on the spatial relations between features of an object are directly expressed in a connectionist network. For two-dimensional shapes, an object instantiation contains 4 degrees of freedom: $(x,y)$-position, orientation, and size. These parameter values, or some recoding of them, can be represented in a set of 4 real-valued *instantiation* units. The network has a modular structure, with units devoted to each object or object fragment to be recognized. In a *recognition module*, one layer of instantiation units represents the instantiation parameters of each of an object's features; these units connect to a set of units that represent the object's instantiation parameters as predicted by this feature; and these predictions are combined into a single object instantiation in another set of instantiation units. The set of weights connecting the instantiation units of the feature and its predicted instantiation for the object are meant to capture the fixed, *linear* reference frame transformation between the feature and the object. These weights are trained by showing various instantiations of the object, and the object's instantiation parameters act as the training signal for each of the features' predictions. Through this supervised procedure, the features of an object learn to predict the instantiation parameters for the object. Thus, when the features of the object are present in the image in the appropriate relationship, the predictions are consistent and this consistency can be used to decide that the object is present. Our simulations showed that TRAFFIC was able to learn to recognize constellations in realistic star-plot images.

### 2.2    LEARNING THE CONSTRAINT: UNSUPERVISED

The goal of the current work is to use an *unsupervised* procedure to discover and use the consistency constraint.

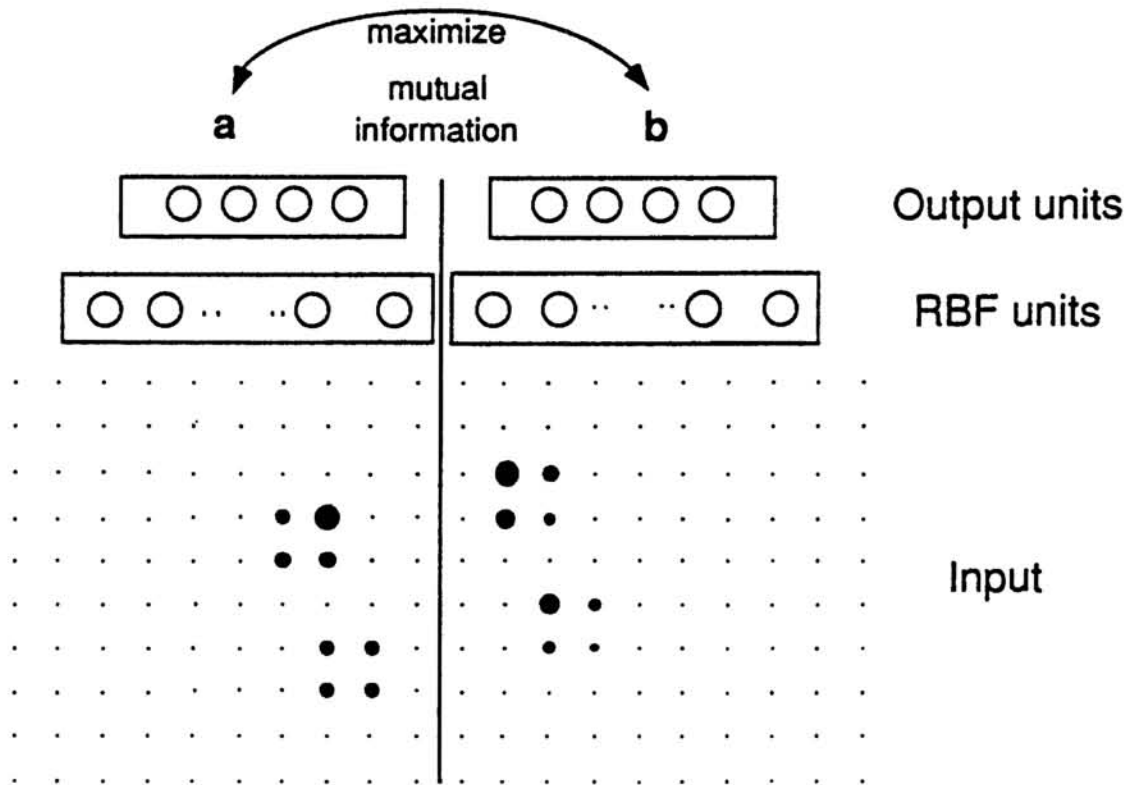

Figure 1: A module with two halves that try to agree on their predictions. The input to each half is 100 intensity values (indicated by the areas of the black circles). Each half has 200 Gaussian radial basis units (constrained to be the same for the two halves) connected to 4 output units.

We explore this idea using a framework similar to that of TRAFFIC, in which different features of an object are represented in different parts of the recognition module, and each part generates a prediction for the object's instantiation parameters. Figure 1 presents an example of the kind of task we would like to solve. The module has two halves. The rigid object in the image is very simple – it has two ends, each of which is composed of two Gaussian blobs of intensity. Each image in the training set contains one instance of the object. For now, we constrain the instantiation parameters of the object so that the left half of the image always contains one end of the object, and the right half the other end. This way, just based on the end of the object in the input image that it sees, each half of the module can always specify the position, orientation and size of the whole object. The goal is that, after training, for any image containing this object, the output vectors of both halves of the module, a and b, should both represent the same instantiation parameters for the object.

In TRAFFIC, we could use the object's instantiation parameters as a training signal for both module halves, and the features would learn their relation to the object. Now, without providing a training signal, we would like the module to learn that what is consistent across the ensemble of images is the relation between the position, orientation, and size of each end of that object. The two halves of a module trained on a particular shape should produce consistent instantiation parameters for any instance of this object. If the features are related in a different way, then these

predictions should disagree. If the module learns to do this through an unsupervised procedure, it has found a viewpoint-invariant spatial relationship that characterizes the object, and can be used to recognize it.

# 3   THE IMAX LEARNING PROCEDURE

We describe a version of the IMAX learning procedure (Hinton and Becker, 1990) that allows a module to discover the 4 parameters that are consistent between the two halves of each image when it is presented with many different images of the same, rigid object in different positions, orientations and sizes. Because the training cases are all positive examples of the object, each half of the module tries to extract a vector of 4 parameters that significantly agrees with the 4 parameters extracted by the other half. Note that the two halves can agree on each instance by outputting zero on each case, but this agreement would not be *significant*. To agree significantly, each output vector must vary from image to image, but the two output vectors must nevertheless be the same for each image. Under suitable Gaussian assumptions, the significance of the agreement between the two output vectors can be computed by comparing the variances across training cases of the parameters produced by the individual halves of the module with the variances of the differences of these parameters.

We assume that the two output vectors, **a** and **b**, are both noisy versions of the same underlying signal, the correct object instantiation parameters. If we assume that the noise is independent, additive, and Gaussian, the mutual information between the presumed underlying signal and the average of the noisy versions of that signal represented by **a** and **b** is:

$$I(\mathbf{a};\mathbf{b}) = \frac{1}{2}\log\frac{|\sum_{(\mathbf{a}+\mathbf{b})}|}{|\sum_{(\mathbf{a}-\mathbf{b})}|} \tag{1}$$

where $|\sum_{(\mathbf{a}+\mathbf{b})}|$ is the determinant of the covariance matrix of the sum of **a** and **b** (see (Becker and Hinton, 1989) for details). We train a recognition module by setting its weights so as to maximize this objective function. By maximizing the determinant, we are discouraging the components of the vector **a** + **b** from being linearly dependent on one another, and thus assure that the network does not discover the same parameter four times.

# 4   EXPERIMENTAL RESULTS

Using this objective function, we have experimented with different training sets, input representations and network architectures. We discuss two examples here.

In all of the experiments described, we fix the number of output units in each module to be 4, matching the underlying degrees of freedom in the object instantiation parameters. We are in effect telling the recognition module that there are 4 parameters worth extracting from the training ensemble. For some tasks there may be less than 4 parameters. For example, the same learning procedure should be able to capture the lower-dimensional constraints between the parts of objects that

contain internal degrees of freedom in their shape (e.g., scissors), but we have not yet tested this.

The first set of experiments uses training images like Figure 1. The task requires an intermediate layer between the intensity values and the instantiation parameters vector. Each half of the module has 200 non-adaptive, radial basis units. The means of the RBFs are formed by randomly sampling the space of possible images of an end of the object; the variances are fixed. The output units are linear. We maximize the objective function I by adjusting the weights from the radial basis units to the output units, after each full sweep through the training set.

The optimization requires 20 sweeps of a conjugate gradient technique through 1000 training cases. Unfortunately, it is difficult to interpret the outputs of the module, since it finds a nonlinear transform of the object instantiation parameters. But the mutual information is quite high – about 7 bits. After training, the predictions made by the two halves are consistent on new images We measure the consistency in the predictions for an image using a kind of generalized Z-score, which relates the difference between the predictions on a particular case $(d_i)$ to the distribution of this difference across the training set:

$$Z(d_i) = (d_i - \overline{d})^t \sum_d^{-1} (d_i - \overline{d}) \qquad (2)$$

A low Z-score indicates a consistent match. After training, the module produces high Z-scores on images where the same two ends are present, but are in a different relationship than the object on which it was trained. In general, the Z-scores increase smoothly with the degree of perturbation in the relationship between the two ends, indicating that the module has learned the constraint.

In the second set of experiments, we remove an unrealistic constraint on our images – that one end of the object must always fall in one half of the image. Instead we assume that there is a feature-extraction process that finds instances of simple features in the image and passes on to the module a set of parameters describing the position, orientation and spatial extent of each feature. This is a reasonable assumption, since low-level vision is generally good at providing accurate descriptions of simple features that are present in an image (such as edges and corners), and can also specify their locations.

In these experiments, the feature-extraction program finds instances of two features of the letter $y$ – the upper u-shaped curve and the long vertical stroke with a curved tail. The recognition module then tries to extract consistent object instantiation parameters from these feature instantiation parameters by maximizing the same mutual information objective as before.

There are several advantages of this second scheme. The first set of training instances were artificially restricted by the requirement that one end must appear in the left half of the image, and the other in the right half. Now since a separate process is analyzing the entire image to find a feature of a given type, we can use the entire space of possible instantiation parameters in the training set. With the simpler architecture, we can efficiently handle more complex images. In addition, no hidden layer is necessary – the mapping from the features' instantiation parameters to the object's instantiation parameters is linear.

Using this scheme, only twelve sweeps through 1000 training cases are necessary

to optimize the objective function. The speed-up is likely due to the fact that the input is already parameterized in an appropriate form for the extraction of the instantiation parameters. This method also produces robust recognition modules, which reject instances where the relationships between the two input vectors does not match the relationship in the training set. We test this robustness by adding noise of varying magnitudes separately to each component of the input vectors, and measuring the Z-scores of the output vectors. As expected, the agreement between the two outputs of a module degrades smoothly with added noise.

## 5    COMPETITIVE IMAX

We are currently working on extending this idea to handle multiple shapes. The obvious way to do this using modules of the type described above is to force the modules to specialize by training each module separately on images of a particular shape, and then to recognize shapes by giving the image to each module and seeing which module achieves the lowest Z-score. However, this requires supervised training in which the images are labelled by the type of object they contain. We are exploring an entirely unsupervised method in which images are *unlabelled*, and every image is processed by many competing modules.

Each competing module has a *responsibility* for each image that depends on the consistency between the two output vectors of the module. The responsibilities are normalized so that, for each image, they sum to one. In computing the covariances for a particular module in Equation 1, we weight each training case by the module's responsibility for that case. We also compute an overall *mixing proportion*, $\pi_m$, for each module which is just the average of its responsibilities. We extend the objective function I to multiple modules as follows:

$$I^* = \sum_m \pi_m \ I_m(\mathbf{a}; \mathbf{b}) \tag{3}$$

We could compute the relative responsibilities of modules by comparing their Z-scores, but this would lead to a recurrent relationship between the responsibilities and the weights within a module. To avoid this recurrence, we simply store the responsibility of each module for each training case. We optimize $I^*$ by interleaving updates of the weights within each module, with updates of the stored responsibilities. This learning is a sophisticated form of competitive learning. Rather than clustering together input vectors that are close to one another in the input space, the modules cluster together input vectors that share a common spatial relationship between their two halves.

In our experiments, we are using just two modules and an ensemble of images of two different shapes (either a *g* or a *y* in each image). We have found that the system can cluster the images with a little bootstrapping. We initially split the training set into *g*-images and *y*-images, and train up one module for several iterations on one set of images, and the other module on the other set. When we then use a new training set containing 500 images of each shape, and train both modules competitively on the full set, the system successfully learns to separate the images so that the modules each specialize in a particular shape. After the bootstrapping, one module wins on 297 cases of one shape and 206 cases of the other shape. After further learning on

the *unlabelled* mixture of shapes, it wins on 498 cases of one shape and 0 cases of the other.

By making another assumption, that the input images in the training set are *temporally* coherent, we should be able to eliminate the need for the bootstrapping procedure. If we assume that the training images come in runs of one class, and then another, as would be the case if they were a sequence of images of various moving objects, then for each module, we can attempt to maximize the mutual information between the responsibilities it assigns to consecutive training images. We can augment the objective function $I^*$ by adding this temporal coherence term onto the spatial coherence term, and our network should cluster the input set into different shapes while simultaneously learning how to recognize them.

Finally, we plan to extend our model to become a more general recognition system. Since the learning relatively is fast, we should also be able to build a hierarchy of modules that could learn to recognize more complex objects.

## Acknowledgements

We thank Sue Becker and Steve Nowlan for helpful discussions. This research was supported by grants from the Ontario Information Technology Research Center, the Natural Sciences and Engineering Research Council, and Apple Computer, Inc. Hinton is the Noranda Fellow of the Canadian Institute for Advanced Research.

## References

Ballard, D. H. (1981). Generalizing the Hough transform to detect arbitrary shapes. *Pattern Recognition*, 13(2):111–122.

Becker, S. and Hinton, G. E. (1989). Spatial coherence as an internal teacher for a neural network. Technical Report Technical Report CRG-TR-89-7, University of Toronto.

Hinton, G. E. (1981). A parallel computation that assigns canonical object-based frames of reference. In *Proceedings of the 7th International Joint Conference on Artificial Intelligence*, pages 683–685, Vancouver, BC, Canada.

Hinton, G. E. and Becker, S. (1990). An unsupervised learning procedure that discovers surfaces in random-dot stereograms. In *Proceedings of the International Joint Conference on Neural Networks*, volume 1, pages 218–222, Hillsdale, NJ. Erlbaum.

Lowe, D. G. (1985). *Perceptual Organization and Visual Recognition*. Kluwer Academic Publishers, Boston.

Lowe, D. G. (1987). The viewpoint consistency constraint. *International Journal of Computer Vision*, 1:57–72.

Roberts, L. G. (1965). Machine perception of three-dimensional solids. In Tippett, J. T., editor, *Optical and Electro-Optical Information Processing*. MIT Press.

Zemel, R. S., Mozer, M. C., and Hinton, G. E. (1989). TRAFFIC: Object recognition using hierarchical reference frame transformations. In Touretzky, D. S., editor, *Advances in Neural Information Processing Systems 2*, pages 266–273. Morgan Kaufmann, San Mateo, CA.